# Implicit Online Learning with Kernels

**Li Cheng**     **S.V. N. Vishwanathan**
National ICT Australia
li.cheng@nicta.com.au
SVN.Vishwanathan@nicta.com.au

**Dale Schuurmans**
Department of Computing Science
University of Alberta, Canada
dale@cs.ualberta.ca

**Shaojun Wang**
Department of Computer Science and Engineering
Wright State University
shaojun.wang@wright.edu

**Terry Caelli**
National ICT Australia
terry.caelli@nicta.com.au

## Abstract

We present two new algorithms for online learning in reproducing kernel Hilbert spaces. Our first algorithm, ILK (implicit online learning with kernels), employs a new, implicit update technique that can be applied to a wide variety of convex loss functions. We then introduce a bounded memory version, SILK (sparse ILK), that maintains a compact representation of the predictor without compromising solution quality, even in non-stationary environments. We prove loss bounds and analyze the convergence rate of both. Experimental evidence shows that our proposed algorithms outperform current methods on synthetic and real data.

## 1 Introduction

Online learning refers to a paradigm where, at each time $t$, an instance $\boldsymbol{x}_t \in \mathcal{X}$ is presented to a learner, which uses its parameter vector $f_t$ to predict a label. This predicted label is then compared to the true label $y_t$, via a non-negative, piecewise differentiable, convex loss function $L(\boldsymbol{x}_t, y_t, f_t)$. The learner then updates its parameter vector to minimize a *risk functional*, and the process repeats.

Kivinen and Warmuth [1] proposed a generic framework for online learning where the risk functional, $J_t(f)$, to be minimized consists of two terms: a Bregman divergence between parameters $\Delta_{\mathrm{G}}(f, f_t) := G(f) - G(f_t) - \langle f - f_t, \partial_f G(f_t) \rangle$, defined via a convex function $G$, and the instantaneous risk $R(\boldsymbol{x}_t, y_t, f)$, which is usually given by a function of the instantaneous loss $L(\boldsymbol{x}_t, y_t, f)$. The parameter updates are then derived via the principle

$$f_{t+1} = \operatorname*{argmin}_{f} J_t(f) := \operatorname*{argmin}_{f} \{ \Delta_{\mathrm{G}}(f, f_t) + \eta_t R(\boldsymbol{x}_t, y_t, f) \}, \tag{1}$$

where $\eta_t$ is the learning rate. Since $J_t(f)$ is convex, (1) is solved by setting the gradient (or, if necessary, a subgradient) to 0. Using the fact that $\partial_f \Delta_{\mathrm{G}}(f, f_t) = \partial_f G(f) - \partial_f G(f_t)$, one obtains

$$\partial_f G(f_{t+1}) = \partial_f G(f_t) - \eta_t \partial_f R(\boldsymbol{x}_t, y_t, f_{t+1}). \tag{2}$$

Since it is difficult to determine $\partial_f R(\boldsymbol{x}_t, y_t, f_{t+1})$ in closed form, an *explicit* update, as opposed to the above *implicit* update, uses the approximation $\partial_f R(\boldsymbol{x}_t, y_t, f_{t+1}) \approx \partial_f R(\boldsymbol{x}_t, y_t, f_t)$ to arrive at the more easily computable expression [1]

$$\partial_f G(f_{t+1}) = \partial_f G(f_t) - \eta_t \partial_f R(\boldsymbol{x}_t, y_t, f_t). \tag{3}$$

In particular, if we set $G(f) = \frac{1}{2}||f||^2$, then $\Delta_{\mathrm{G}}(f, f_t) = \frac{1}{2}||f - f_t||^2$ and $\partial_f G(f) = f$, and we obtain the familiar stochastic gradient descent update

$$f_{t+1} = f_t - \eta_t \partial_f R(\boldsymbol{x}_t, y_t, f_t). \tag{4}$$

We are interested in applying online learning updates in a reproducing kernel Hilbert space (RKHS). To lift the above update into an RKHS, $\mathcal{H}$, one typically restricts attention to $f \in \mathcal{H}$ and defines [2]

$$R(\boldsymbol{x}_t, y_t, f) := \frac{\lambda}{2}||f||^2_{\mathcal{H}} + C \cdot L(\boldsymbol{x}_t, y_t, f), \tag{5}$$

where $||\cdot||_{\mathcal{H}}$ denotes the RKHS norm, $\lambda > 0$ is a regularization constant, and $C > 0$ determines the penalty imposed on point prediction violations.

Recall that if $\mathcal{H}$ is a RKHS of functions on $\mathcal{X} \times \mathcal{Y}$, then its defining kernel $k : (\mathcal{X} \times \mathcal{Y})^2 \to \mathbb{R}$ satisfies the reproducing property; namely that $\langle f, k((\boldsymbol{x}, y), \cdot) \rangle_{\mathcal{H}} = f(\boldsymbol{x}, y)$ for all $f \in \mathcal{H}$. Therefore, by making the standard assumption that $L$ only depends on $f$ via its evaluations at $f(\boldsymbol{x}, y)$, one reaches the conclusion that $\partial_f L(\boldsymbol{x}, y, f) \in \mathcal{H}$, and in particular

$$\partial_f L(\boldsymbol{x}, y, f) = \sum_{\tilde{y} \in \mathcal{Y}} \beta_{\tilde{y}} k((\boldsymbol{x}, \tilde{y}), \cdot), \tag{6}$$

for some $\beta_{\tilde{y}} \in \mathbb{R}$. Since $\partial_f R(\boldsymbol{x}_t, y_t, f_t) = \lambda f_t + C \cdot \partial_f L(\boldsymbol{x}_t, y_t, f_t)$, one can use (4) to obtain an explicit update $f_{t+1} = (1 - \eta_t \lambda) f_t - \eta_t C \cdot \partial_f L(\boldsymbol{x}_t, y_t, f_t)$, which combined with (6) shows that there must exist coefficients $\alpha_{i,\tilde{y}}$ fully specifying $f_{t+1}$ via

$$f_{t+1} = \sum_{i=1}^{t} \sum_{\tilde{y} \in \mathcal{Y}} \alpha_{i,\tilde{y}} k((\boldsymbol{x}_i, \tilde{y}), \cdot). \tag{7}$$

In this paper we propose an algorithm ILK (implicit online learning with kernels) that solves (2) directly, while still expressing updates in the form (7). That is, we derive a technique for computing the *implicit* update that can be applied to many popular loss functions, including quadratic, hinge, and logistic losses, as well as their extensions to structured domains (see *e.g.* [3])—in an RKHS. We also provide a general recipe to check if a new convex loss function is amenable to these implicit updates. Furthermore, to reduce the memory requirement of ILK, which grows linearly with the number of observations (instance-label pairs), we propose a sparse variant SILK (sparse ILK) that approximates the decision function $f$ by truncating past observations with insignificant weights.

## 2 Implicit Updates in an RKHS

As shown in (1), to perform an implicit update one needs to minimize $\Delta_{\mathrm{G}}(f, f_t) + R(\boldsymbol{x}_t, y_t, f)$. By replacing $R(\boldsymbol{x}_t, y_t, f)$ with (5), and setting $G(f) = \frac{1}{2}||f||_{\mathcal{H}}^2$, one obtains

$$f_{t+1} = \arg\min_f J(f) = \operatorname*{argmin}_f \; \frac{1}{2}||f - f_t||_{\mathcal{H}}^2 + \eta_t \left( \frac{\lambda}{2}||f||_{\mathcal{H}}^2 + C \cdot L(\boldsymbol{x}_t, y_t, f) \right). \tag{8}$$

Since $L$ is assumed convex with respect to $f$, setting $\partial_f J = 0$ and using an auxiliary variable $\tau_t = \frac{\eta_t \lambda}{1 + \eta_t \lambda}$ yields

$$f_{t+1} = (1 - \tau_t) f_t - (1 - \tau_t) \eta_t C \partial_f L(\boldsymbol{x}_t, y_t, f_{t+1}). \tag{9}$$

On the other hand, from the form (7) it follows that $f_{t+1}$ can also be written as

$$f_{t+1} = \sum_{i=1}^{t-1} \sum_{\tilde{y} \in \mathcal{Y}} \alpha_{i,\tilde{y}} k((\boldsymbol{x}_i, \tilde{y}), \cdot) + \sum_{\tilde{y} \in \mathcal{Y}} \alpha_{t,\tilde{y}} k((\boldsymbol{x}_t, \tilde{y}), \cdot), \tag{10}$$

for some $\alpha_{j,\tilde{y}} \in \mathbb{R}$ and $j = 1, \dots, t$. Since

$$\partial_f L(\boldsymbol{x}_t, y_t, f_{t+1}) = \sum_{\tilde{y} \in \mathcal{Y}} \beta_{t,\tilde{y}} k((\boldsymbol{x}_t, \tilde{y}), \cdot),$$

and for ease of exposition, we assume a fixed step size (learning rate) $\eta_t = 1$, consequently $\tau_t = \tau$, it follows from (9) and (10) that

$$\alpha_{i,\tilde{y}} = (1 - \tau) \alpha_{i,\tilde{y}} \qquad \text{for } i = 1, \dots, t-1, \text{ and } \tilde{y} \in \mathcal{Y}, \tag{11}$$

$$\alpha_{t,\tilde{y}} = -(1 - \tau) C \beta_{t,\tilde{y}} \quad \text{for all } \tilde{y} \in \mathcal{Y}. \tag{12}$$

Note that sophisticated step size adaptation algorithms (*e.g.* [3]) can be modified in a straightforward manner to work in our setting.

The main difficulty in performing the above update arises from the fact that $\beta_{t,\tilde{y}}$ depends on $f_{t+1}$ (see *e.g.* (13)) which in turn depends on $\beta_{t,\tilde{y}}$ via $\alpha_{t,\tilde{y}}$. The general recipe to overcome this problem is to first use (9) to write $\beta_{t,\tilde{y}}$ as a function of $\alpha_{t,\tilde{y}}$. Plugging this back into (12) yields an equation in $\alpha_{t,\tilde{y}}$ alone, which sometimes can be solved efficiently. We now elucidate the details for some well-known loss functions.

**Square Loss** In this case, $k((\boldsymbol{x}_t, y_t), \cdot) = k(\boldsymbol{x}_t, \cdot)$. That is, the kernel does not depend on the value of $y$. Furthermore, we assume that $\mathcal{Y} = \mathbb{R}$, and write

$$L(\boldsymbol{x}_t, y_t, f) = \frac{1}{2}(f(\boldsymbol{x}_t) - y_t)^2 = \frac{1}{2}(\langle f(\cdot), k(\boldsymbol{x}_t, \cdot)\rangle_{\mathcal{H}} - y_t)^2,$$

which yields

$$\partial_f L(\boldsymbol{x}_t, y_t, f) = (f(\boldsymbol{x}_t) - y_t)\, k(\boldsymbol{x}_t, \cdot). \tag{13}$$

Substituting into (12) and using (9) we have

$$\alpha_t = -(1 - \tau)C((1 - \tau)f_t(\boldsymbol{x}_t) + \alpha_t k(\boldsymbol{x}_t, \boldsymbol{x}_t) - y_t).$$

After some straightforward algebraic manipulation we obtain the solution

$$\alpha_t = \frac{C(1 - \tau)(y_t - (1 - \tau)f_t(\boldsymbol{x}_t))}{1 + C(1 - \tau)k(\boldsymbol{x}_t, \boldsymbol{x}_t)}.$$

**Binary Hinge Loss** As before, we assume $k((\boldsymbol{x}_t, y_t), \cdot) = k(\boldsymbol{x}_t, \cdot)$, and set $\mathcal{Y} = \{\pm 1\}$. The hinge loss for binary classification can be written as

$$L(\boldsymbol{x}_t, y_t, f) = (\rho - y_t f(\boldsymbol{x}_t))_+ = (\rho - y_t \langle f, k(\boldsymbol{x}_t, \cdot)\rangle_{\mathcal{H}})_+, \tag{14}$$

where $\rho > 0$ is the margin parameter, and $(\cdot)_+ := \max(0, \cdot)$. Recall that the subgradient is a set, and the function is said to be differentiable at a point if this set is a singleton [4]. The binary hinge loss is not differentiable at the hinge point, but its subgradient exists everywhere. Writing $\partial_f L(\boldsymbol{x}_t, y_t, f) = \beta_t k(\boldsymbol{x}_t, \cdot)$ we have:

$$y_t f(\boldsymbol{x}_t) > \rho \implies \beta_t = 0; \tag{15a}$$
$$y_t f(\boldsymbol{x}_t) = \rho \implies \beta_t \in [0, -y_t]; \tag{15b}$$
$$y_t f(\boldsymbol{x}_t) < \rho \implies \beta_t = -y_t. \tag{15c}$$

We need to balance between two conflicting requirements while computing $\alpha_t$. On one hand we want the loss to be zero, which can be achieved by setting $\rho - y_t f_{t+1}(\boldsymbol{x}_t) = 0$. On the other hand, the gradient of the loss at the new point $\partial_f L(\boldsymbol{x}_t, y_t, f_{t+1})$ must satisfy (15). We satisfy both constraints by appropriately clipping the *optimal* estimate of $\alpha_t$.

Let $\hat{\alpha}_t$ denote the optimal estimate of $\alpha_t$ which leads to $\rho - y_t f_{t+1}(\boldsymbol{x}_t) = 0$. Using (9) we have $\rho - y_t\left((1 - \tau)f_t(\boldsymbol{x}_t) + \hat{\alpha}_t k(\boldsymbol{x}_t, \boldsymbol{x}_t)\right) = 0$, which yields

$$\hat{\alpha}_t = \frac{\rho - (1 - \tau)y_t f_t(\boldsymbol{x}_t)}{y_t k(\boldsymbol{x}_t, \boldsymbol{x}_t)} = \frac{y_t(\rho - (1 - \tau)y_t f_t(\boldsymbol{x}_t))}{k(\boldsymbol{x}_t, \boldsymbol{x}_t)}.$$

On the other hand, by using (15) and (12) we have $\alpha_t y_t \in [0, (1 - \tau)C]$. By combining the two scenarios, we arrive at the final update

$$\alpha_t = \begin{cases} \hat{\alpha}_t & \text{if } y_t \hat{\alpha}_t \in [0, (1 - \tau)C]; \\ 0 & \text{if } y_t \hat{\alpha}_t < 0; \\ y_t(1 - \tau)C & \text{if } y_t \hat{\alpha}_t > (1 - \tau)C. \end{cases} \tag{16}$$

The updates for the hinge loss used in novelty detection are very similar.

**Graph Structured Loss** The graph-structured loss on label domain can be written as

$$L(\boldsymbol{x}_t, y_t, f) = \left(-f(\boldsymbol{x}_t, y_t) + \max_{\tilde{y} \neq y_t}(\Delta(y_t, \tilde{y}) + f(\boldsymbol{x}_t, \tilde{y}))\right)_+. \tag{17}$$

Here, the margin of separation between labels is given by $\Delta(y_t, \tilde{y})$ which in turn depends on the graph structure of the output space. This a very general loss, which includes binary and multiclass hinge loss as special cases (see *e.g.* [3]). We briefly summarize the update equations for this case.

Let $y^* = \operatorname{argmax}_{\tilde{y} \neq y_t}\{\Delta(y_t, \tilde{y}) + f_t(\boldsymbol{x}_t, \tilde{y})\}$ denote the best runner-up label for current instance $x_t$. Then set $\alpha_{t, y_t} = -\alpha_{t, y^*} = \alpha_t$, use $k_t(y, y')$ to denote $k((x_t, y), (x_t, y'))$ and write

$$\hat{\alpha}_t = \frac{-(1 - \tau)f_t(x_t, y_t) + \Delta(y_t, y^*) + (1 - \tau)f_t(x_t, y^*)}{(k_t(y_t, y_t) + k_t(y^*, y^*) - 2k_t(y_t, y^*))}.$$

The updates are now given by

$$\alpha_t = \begin{cases} 0 & \text{if } \hat{\alpha}_t < 0; \\ \hat{\alpha}_t & \text{if } \hat{\alpha}_t \in [0, (1-\tau)C]; \\ (1-\tau)C & \text{if } \hat{\alpha}_t > (1-\tau)C. \end{cases} \qquad (18)$$

**Logisitic Regression Loss**   The logistic regression loss and its gradient can be written as

$$L(\boldsymbol{x}_t, y_t, f) = \log\left(1 + \exp(-y_t f(\boldsymbol{x}_t))\right), \quad \partial_f L(\boldsymbol{x}_t, y_t, f) = \frac{-y_t k(\boldsymbol{x}_t, \cdot)}{1 + \exp(y_t f(\boldsymbol{x}_t))}.$$

respectively. Using (9) and (12), we obtain

$$\alpha_t = \frac{(1-\tau)C y_t}{1 + \exp(y_t(1-\tau)f_t(\boldsymbol{x}_t) + \alpha_t y_t k(\boldsymbol{x}_t, \boldsymbol{x}_t))}.$$

Although this equation does not give a closed-form solution, the value of $\alpha_t$ can still be obtained by using a numerical root-finding routine, such as those described in [5].

## 2.1 ILK and SILK Algorithms

We refer to the algorithm that performs implicit updates as ILK, for "implicit online learning with kernels". The update equations of ILK enjoy certain advantages. For example, using (11) it is easy to see that an exponential decay term can be naturally incorporated to down-weight past observations:

$$f_{t+1} = \sum_{i=1}^{t} \sum_{\tilde{y} \in \mathcal{Y}} (1-\tau)^{t-i} \alpha_{i, \tilde{y}} k((\boldsymbol{x}_i, \tilde{y}), \cdot). \qquad (19)$$

Intuitively, the parameter $\tau \in (0, 1)$ (determined by $\lambda$ and $\eta$) trades off between the regularizer and the loss on the current sample. In the case of hinge losses—both binary and graph structured—the weight $|\alpha_t|$ is always upper bounded by $(1-\tau)C$, which ensures limited influence from outliers (*cf.* (16) and (18)).

A major drawback of the ILK algorithm described above is that the size of the kernel expansion grows linearly with the number of data points up to time $t$ (see (10)). In many practical domains, where real time prediction is important (for example, video surveillance), storing all the past observations and their coefficients is prohibitively expensive. Therefore, following Kivinen et al. [2] and Vishwanathan et al. [3] one can truncate the function expansion by storing only a few relevant past observations. We call this version of our algorithm SILK, for "sparse ILK".

Specifically, the SILK algorithm maintains a buffer of size $\omega$. Each new point is inserted into the buffer with coefficient $\alpha_t$. Once the buffer limit $\omega$ is exceeded, the point with the lowest coefficient value is discarded to maintain a bound on memory usage. This scheme is more effective than the straightforward *least recently used* (LRU) strategy proposed in Kivinen et al. [2] and Vishwanathan et al. [3]. It is relatively straightforward to show that the difference between the true predictor and its truncated version obtained by storing only $\omega$ expansion coefficients decreases exponentially as the buffer size $\omega$ increases [2].

# 3   Theoretical Analysis

In this section we will primarily focus on analyzing the graph-structured loss (17), establishing relative loss bounds and analyzing the rate of convergence of ILK and SILK. Our proof techniques adopt those of Kivinen et al. [2]. Due to the space constraints, we leave some details and analysis to the full version of the paper. Although the bounds we obtain are similar to those obtained in [2], our experimental results clearly show that ILK and SILK are stronger than the NORMA strategy of [2] and its truncated variant.

## 3.1   Mistake Bound
We begin with a technical definition.

**Definition 1** *A sequence of hypotheses* $\{(f_1, \ldots, f_T) : f_t \in \mathcal{H}\}$ *is said to be* $(T, B, D_1, D_2)$ bounded *if it satisfies* $||f_t||_{\mathcal{H}}^2 \leq B^2 \ \forall t \in \{1, \ldots, T\}$, $\sum_t ||f_t - f_{t+1}||_{\mathcal{H}} \leq D_1$, *and* $\sum_t ||f_t - f_{t+1}||_{\mathcal{H}}^2 \leq D_2$ *for some* $B, D_1, D_2 \geq 0$. *The set of all* $(T, B, D_1, D_2)$ *bounded hypothesis sequences is denoted as* $\mathcal{F}(T, B, D_1, D_2)$.

Given a fixed sequence of observations $\{(\boldsymbol{x}_1, y_1), \ldots, (\boldsymbol{x}_T, y_T)\}$, and a sequence of hypotheses $\{(f_1, \ldots, f_T) \in \mathcal{F}\}$, the number of errors $M$ is defined as

$$M := |\{t : \Delta f(\boldsymbol{x}_t, y_t, y_t^*) \leq 0\}|,$$

where $\Delta f(\boldsymbol{x}_t, y_t, y_t^*) = f(\boldsymbol{x}_t, y_t) - f(\boldsymbol{x}_t, y_t^*)$ and $y_t^*$ is the best runner-up label. To keep the equations succinct, we denote $\Delta k_t((y_t, y), \cdot) := k((\boldsymbol{x}_t, y_t), \cdot) - k((\boldsymbol{x}_t, y), \cdot)$, and $\Delta k_t((y_t, y), (y_t, y)) := \|\Delta k_t((y_t, y), \cdot)\|_{\mathcal{H}}^2 = k_t(y_t, y_t) - 2k_t(y_t, y) + k_t(y, y)$. In the following we bound the number of mistakes $M$ made by ILK by the cumulative loss of an arbitrary sequence of hypotheses from $\mathcal{F}(T, B, D_1, D_2)$.

**Theorem 2** *Let $\{(\boldsymbol{x}_1, y_1), \ldots, (\boldsymbol{x}_T, y_T)\}$ be an arbitrary sequence of observations such that $\Delta k_t((y_t, y), (y_t, y)) \leq X^2$ holds for any $t$, any $y$, and for some $X > 0$. For an arbitrary sequence of hypotheses $(g_1, \cdots, g_T) \in \mathcal{F}(T, B, D_1, D_2)$ with average margin $\mu = \frac{1}{|\mathcal{E}|} \sum_{t \in \mathcal{E}} \left( \Delta(y_t, y_t^{g*}) - \Delta(y_t, y_t^*) \right)$, and bounded cumulative loss $K := \sum_t L(\boldsymbol{x}_t, y_t, g_t)$, the number of mistakes of the sequence of hypotheses $(f_1, \cdots, f_T)$ generated by ILK with learning rate $\eta_t = \eta$, $\lambda = \frac{1}{B\eta}\sqrt{\frac{D_2}{T}}$ is upper-bounded by*

$$M \leq \frac{K}{\mu} + \frac{2S}{\mu^2} + 2\left(\frac{K}{\mu} + \frac{S}{\mu^2}\right)^{\frac{1}{2}} \left(\frac{S}{\mu^2}\right)^{\frac{1}{2}}, \tag{20}$$

*where $S = \frac{X^2}{4}(B^2 + BD_1 + B\sqrt{TD_2})$, $\mu > 0$, and $y_t^{g*}$ denotes the best runner-up label with hypothesis $g_t$.*

When considering the stationary distribution in a separable (noiseless) scenario, this theorem allows us to obtain a mistake bound that is reminiscent of the Perceptron convergence theorem. In particular, if we assume the sequence of hypotheses $(g_1, \cdots, g_T) \in \mathcal{F}(T, B, D_1 = 0, D_2 = 0)$ and the cumulative loss $K = 0$, we obtain a bound on the number of mistakes

$$M \leq \frac{B^2 X^2}{\mu^2}. \tag{21}$$

### 3.2 Convergence Analysis

The following theorem asserts that under mild assumptions, the cumulative risk $\sum_{t=1}^{T} R(\boldsymbol{x}_t, y_t, f_t)$ of the hypothesis sequence produced by ILK converges to the minimum risk of the batch learning counterpart $g^* := \text{argmin}_{g \in \mathcal{H}} \sum_{t=1}^{T} R(\boldsymbol{x}_t, y_t, g)$ at a rate of $O(T^{-1/2})$.

**Theorem 3** *Let $\{(\boldsymbol{x}_1, y_1), \ldots, (\boldsymbol{x}_T, y_T)\}$ be an arbitrary sequence of observations such that $\Delta k_t((y_t, y_t), (y_t, y_t)) \leq X^2$ holds for any $t$, any $y$. Denote $(f_1, \ldots, f_T)$ the sequence of hypotheses produced by ILK with learning rate $\eta_t = \eta t^{-1/2}$, $\sum_{t=1}^{T} R(\boldsymbol{x}_t, y_t, f_t)$ the cumulative risk of this sequence, and $\sum_{t=1}^{T} R(\boldsymbol{x}_t, y_t, g)$ the batch cumulative risk of $(g, \ldots, g)$, for any $g \in \mathcal{H}$. Then*

$$\sum_{t=1}^{T} R(\boldsymbol{x}_t, y_t, f_t) \leq \sum_{t=1}^{T} R(\boldsymbol{x}_t, y_t, g) + aT^{1/2} + b,$$

*where $U = \frac{CX}{\lambda}$, $a = 4\eta C^2 X^2 + \frac{2U^2}{\eta}$, and $b = \frac{U^2}{2\eta}$ are constants. In particular, if*

$$g^* = \arg\min_{g \in \mathcal{H}} \sum_{t=1}^{T} R(\boldsymbol{x}_t, y_t, g),$$

*we obtain*

$$\frac{1}{T}\sum_{t=1}^{T} R(\boldsymbol{x}_t, y_t, f_t) \leq \frac{1}{T}\sum_{t=1}^{T} R(\boldsymbol{x}_t, y_t, g^*) + O(T^{-1/2}). \tag{22}$$

Essentially the same theorem holds for SILK, but now with a slightly larger constant $a = 4\eta(1 + \frac{2}{\lambda})C^2 X^2 + \frac{2U^2}{\eta}$. In addition, denote $g^*$ the minimizer of the batch learning cumulative risk $\sum_t R(\boldsymbol{x}_t, y_t, g)$, and $f^*$ the minimizer of the minimum expected risk with $R(f^*) := \min_f \mathbb{E}_{(\boldsymbol{x}, y) \sim P(\boldsymbol{x}, y)} R(\boldsymbol{x}, y, f)$. As stated in [6] for the structured risk minimization framework, as

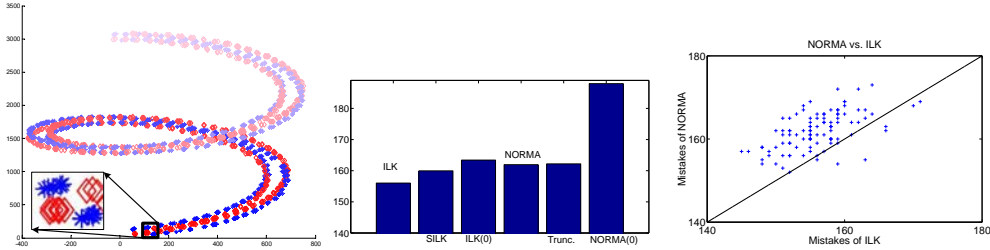

**Figure 1:** The left panel depicts a synthetic data sequence containing two classes (blue crosses and red diamonds, see the zoomed-in portion in bottom-left corner), with each class being sampled from a mixture of two drifting Gaussian distributions. Performance comparison of ILK vs NORMA and truncated NORMA on this data: Average cumulative error over 100 trials (middle), and average cumulative error each trial (right).

the sample size $T$ grows, $T \rightarrow \infty$, we obtain $g^* \rightarrow f^*$ in probability. This subsequently guarantees the convergence of the average regularized risk of ILK and SILK to $R(f^*)$.

The upper bound in the above theorem can be directly plugged into Corollary 2 of Cesa-Bianchi et al. [7] to obtain bounds on the generalization error of ILK. Let $\bar{f}$ denote the average hypothesis produced by averaging over all hypotheses $f_1, \ldots, f_T$. Then for any $\delta \in (0, 1)$, with probability at least $1 - \delta$, the expected risk of $\bar{f}$ is upper bounded by the risk of the best hypothesis chosen in hindsight plus a term which grows as $O\left(\sqrt{\frac{1}{T}}\right)$.

## 4 Experiments

We evaluate the performance of ILK and SILK by comparing them to NORMA [2] and its truncated variant. On OCR data, we also compare our algorithms to SVMD, a sophisticated step-size adaptation algorithm in RKHS presented in [3]. For a fair comparison we tuned the parameters of each algorithm separately and report the best results. In addition, we fixed the margin to $\rho = 1$ for all our loss functions.

**Binary Classification on Synthetic Sequences**    The aim here is to demonstrate that ILK is better than NORMA in coping with non-stationary distributions. Each trial of our experiment works with 2000 two-dimensional instances sampled from a non-stationary distribution (see Figure 1) and the task is to classify the sampled points into one of two classes. The central panel of Figure 1 compares the number of errors made by various algorithms, averaged over 100 trials. Here, ILK and SILK make fewer mistakes than NORMA and truncated NORMA. We also tested two other algorithms, ILK(0) obtained by setting the decay factor $\lambda$ to zero, and similarly for NORMA(0). As expected, both these variants make more mistakes because they are unable to forget the past, which is crucial for obtaining good performance in a non-stationary environment. To further compare the performance of ILK and NORMA we plot the relative errors of these two algorithms in the right panel of Figure 1. As can be seen, ILK out-performs NORMA on this simple non-stationary problem.

**Novelty Detection on Video Sequences**    As a significant application, we applied SILK to a background subtraction problem in video data analysis. The goal is to detect the moving foreground objects (such as cars, persons, etc) from relatively static background scenes in real time. The challenge in this application is to be able to cope with variations in lighting as well as jitter due to shaking of the camera. We formulate the problem as a novelty detection task using a network of classifiers, one for each pixel. For this task we compare the performance of SILK vs. truncated NORMA. (The ILK and NORMA algorithms are not suitable since their storage requirements grow linearly). A constant buffer size $\omega = 20$ is used for both algorithms in this application. We report further implementation details in the full version of this paper.

The first task is to identify people, under varying lighting conditions, in an indoor video sequence taken with a static camera. The left hand panel of Figure 2 plots the ROC curves of NORMA and SILK, which demonstrates the overall better performance of SILK. We sampled one of the initial frames after the light was switched off and back on. The results are shown in the right panel of Figure 2. As can be seen, SILK is able to recover from the change in lighting condition better than NORMA, and is able to identify foreground objects reasonably close to the ground truth.

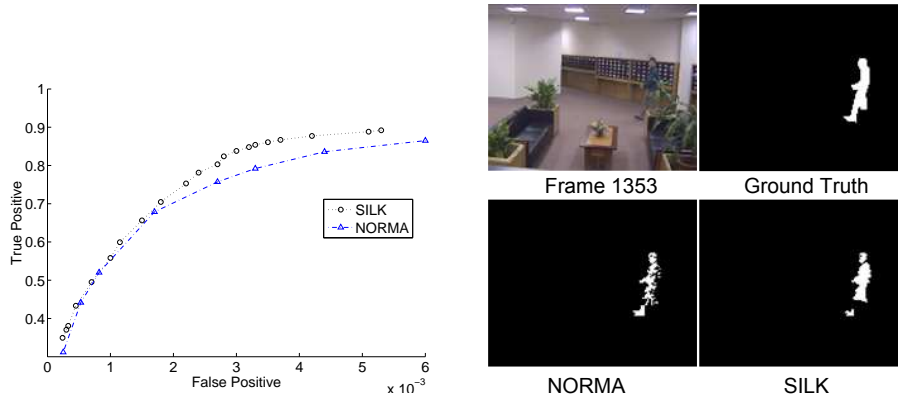

**Figure 2:** Performance comparison of SILK vs truncated NORMA on a background subtraction (moving object detection) task, with varying lighting conditions. ROC curve (left) and a comparison of algorithms immediately after the lights have been switched off and on (right).

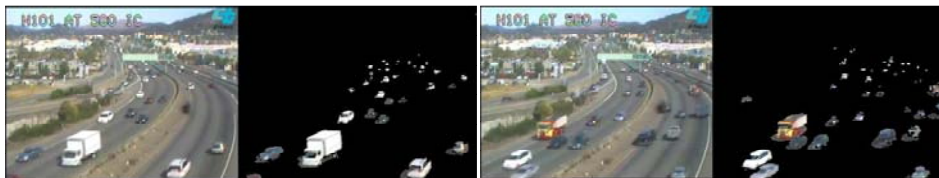

**Figure 3:** Performance of SILK on a road traffic sequence (moving car detection) task, with a jittery camera. Two random frames and the performance of SILK on those frames are depicted.

Our second experiment is a traffic sequence taken by a camera that shakes irregularly, which creates a challenging problem for any novelty detection algorithm. As seen from the randomly chosen frames plotted in Figure 3 SILK manages to obtain a visually plausible detection result. We cannot report a quantitative comparison with other methods in this case, due to the lack of manually labeled ground-truth data.

**Binary and Multiclass Classification on OCR data** We present two sets of experiments on the MNIST dataset. The aim of the first set experiment is to show that SILK is competitive with NORMA and SVMD on a simple binary task. The data is split into two classes comprising the digits $0 - 4$ and $5 - 9$, respectively. A polynomial kernel of degree 9 and a buffer size of $\omega = 128$ is employed for all three algorithms. Figure 4 (a) plots current average error rate, *i.e.,* the total number of errors on the examples seen so far divided by the iteration number. As can be seen, after the initial oscillations have died out, SILK consistently outperforms SVMD and NORMA, achieving a lower average error after one pass through the dataset. Figure 4 (b) examines the effect of buffer size on SILK. As expected, smaller buffer sizes result in larger truncation error and hence worse performance. With increasing buffer size the asymptotic average error decreases. For the 10-way multiclass classification task we set $\omega = 128$, and used a Gaussian kernel following [3]. Figure 4 (c) shows that SILK consistently outperforms NORMA and SVMD, while the trend with the increasing buffer size is repeated, as shown in Figure 4 (d). In both experiments, we used the parameters for NORMA and SVMD reported in [3], and set $\tau = 0.00005$ and $C = 100$ for SILK.

## 5   Outlook and Discussion

In this paper we presented a general recipe for performing implicit online updates in an RKHS. Specifically, we showed that for many popular loss functions these updates can be computed efficiently. We then presented a sparse version of our algorithm which uses limited basis expansions to approximate the function. For graph-structured loss we also showed loss bounds and rates of convergence. Experiments on real life datasets demonstrate that our algorithm is able to track nonstationary targets, and outperforms existing algorithms.

For the binary hinge loss, when $\tau = 0$ the proposed update formula for $\alpha_t$ (16) reduces to the PA-I algorithm of Crammer et al. [8]. Curiously enough, the motivation for the updates in both cases seems completely different. While we use an implicit update formula Crammer et al. [8] use

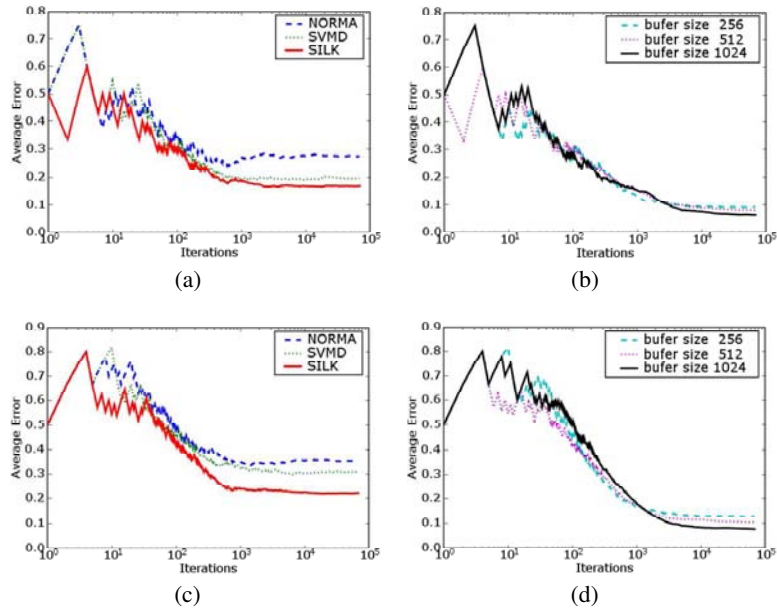

**Figure 4:** Performance comparison of different algorithms over one run of the MNIST dataset. (a) Online binary classification. (b) Performance of SILK using different buffer sizes. (c) Online 10-way multiclass classification. (d) Performance of SILK on three different buffer sizes.

a Lagrangian formulation, and a passive-aggressive strategy. Furthermore, the loss functions they handle are generally linear (hinge loss and its various generalizations) while our updates can handle other non-linear losses such as quadratic or logistic loss.

Our analysis of loss bounds is admittedly straightforward given current results. The use of more sophisticated analysis and extending our bounds to deal with other non-linear loss functions is on-going. We are also applying our techniques to video analysis applications by exploiting the structure of the output space.

### Acknowledgements

We thank Xinhua Zhang, Simon Guenter, Nic Schraudolph and Bob Williamson for carefully proof reading the paper, pointing us to many references, and helping us improving presentation style. National ICT Australia is funded by the Australian Government's Department of Communications, Information Technology and the Arts and the Australian Research Council through Backing Australia's Ability and the ICT Center of Excellence program. This work is supported by the IST Program of the European Community, under the Pascal Network of Excellence, IST-2002-506778.

### References

[1] J. Kivinen and M. K. Warmuth. Exponentiated gradient versus gradient descent for linear predictors. *Information and Computation*, 132(1):1–64, 1997.

[2] J. Kivinen, A. J. Smola, and R. C. Williamson. Online learning with kernels. *IEEE Transactions on Signal Processing*, 52(8), 2004.

[3] S. V. N. Vishwanathan, N. N. Schraudolph, and A. J. Smola. Step size adaptation in reproducing kernel Hilbert space. *Journal of Machine Learning Research*, 7, 2006.

[4] R. T. Rockafellar. *Convex Analysis*, volume 28 of *Princeton Mathematics Series*. Princeton University Press, 1970.

[5] W. H. Press, S. A. Teukolsky, W. T. Vetterling, and B. P. Flannery. *Numerical Recipes in C: The Art of Scientific Computing (2nd ed.)*. Cambridge University Press, Cambridge, 1992. ISBN 0 - 521 - 43108 - 5.

[6] V. Vapnik. *Statistical Learning Theory*. John Wiley and Sons, New York, 1998.

[7] N. Cesa-Bianchi, A. Conconi, and C. Gentile. On the generalization ability of on-line learning algorithms. *IEEE Trans. Information Theory*, 50(9):2050–2057, 2004.

[8] K. Crammer, O. Dekel, J. Keshet, S. Shalev-Shwartz, and Y. Singer. Online passive-aggressive algorithms. *Journal of Machine Learning Research*, 7:551–585, 2006.
